# Multi-Layer Perceptrons
# with B-Spline Receptive Field Functions

Stephen H. Lane, Marshall G. Flax, David A. Handelman and Jack J. Gelfand

Human Information Processing Group
Department of Psychology
Princeton University
Princeton, New Jersey 08544

## ABSTRACT

Multi-layer perceptrons are often slow to learn nonlinear functions with complex local structure due to the global nature of their function approximations. It is shown that standard multi-layer perceptrons are actually a special case of a more general network formulation that incorporates B-splines into the node computations. This allows novel spline network architectures to be developed that can combine the generalization capabilities and scaling properties of *global* multi-layer feedforward networks with the computational efficiency and learning speed of *local* computational paradigms. Simulation results are presented for the well known spiral problem of Weiland and of Lang and Witbrock to show the effectiveness of the Spline Net approach.

## 1. INTRODUCTION

Recently, it has been shown that multi-layer feedforward neural networks, such as Multi-Layer Perceptrons (MLPs), are theoretically capable of representing arbitrary mappings, provided that a sufficient number of units are included in the hidden layers (Hornik et al., 1989). Since all network weights are updated with each training exemplar, these networks construct *global* approximations to multi-input/multi-output function data in a manner analogous to fitting a low-order polynomial through a set of

data points. This is illustrated by the cubic polynomial "Global Fit" of the data points in Fig. 1.

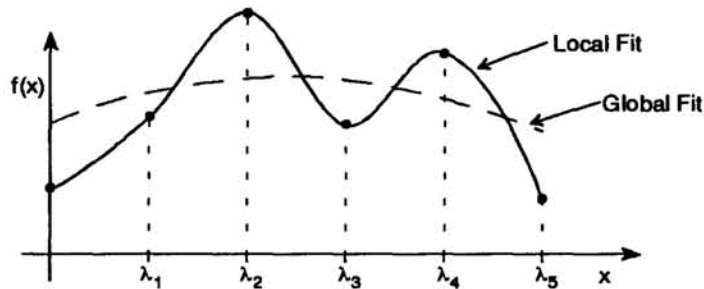

Figure 1.   Global vs. Local Function Approximation

Consequently, multi-layer perceptrons are capable of generalizing (extrapolating/ interpolating) their response to regions of the input space where little or no training data is present, using a quantity of connection weights that typically scales quadratically with the number of hidden nodes.  The global nature of the weight updating, however, tends to blur the details of local structures, slows the rate of learning, and makes the accuracy of the resulting function approximation sensitive to the order of presentation of the training data.

It is well known that many sensorimotor structures in the brain are organized using neurons that possess locally-tuned overlapping receptive fields (Hubel and Wiesel, 1962).  Several neural network computational paradigms such as CMACs (Cerebellar Model Articulation Controllers) (Albus, 1973) and Radial Basis Functions (RBFs) (Moody and Darken, 1988) have been quite successful representing complex nonlinear functions using this same organizing principle.   These networks construct *local* approximations to multi-input/multi-output function data that are analogous to fitting a least-squares spline through a set of data points using piecewise polynomials or other basis functions. This is illustrated as the cubic spline "Local Fit" in Fig. 1.   The main benefits of using local approximation techniques to represent complex nonlinear functions include fast learning and reduced sensitivity to the order of presentation of training data.  In many cases, however, in order to represent the function to the desired degree of smoothness, the number of basis functions required to adequately span the input space can scale exponentially with the number of inputs (Lane et al., 1991a,b).

The work presented in this paper is part of a larger effort (Lane et al, 1991a) to develop a general neural network formulation that can combine the generalization capabilities and scaling properties of *global* multi-layer feedforward networks with the computational efficiency and learning speed of *local* network paradigms.  It is shown in the sequel that this can be accomplished by incorporating B-Spline receptive fields into the node connection functions of Multi-Layer Perceptrons.

## 2.   MULTI-LAYER PERCEPTRONS
## WITH B-SPLINE RECEPTIVE FIELD FUNCTIONS

Standard Multi-Layer Perceptrons (MLPs) can be represented using node equations of the form,

$$y_i^L = \sigma(\sum_{j=0}^{\eta_{L-1}} c_{ij}^L) = \frac{1}{1 + \exp(-\sum_{j=0}^{\eta_{L-1}} c_{ij}^L)} \qquad (1)$$

where $\eta_L$ is the number of nodes in layer $L$ and the $c_{ij}^L$ are linear connection functions between nodes in layers $L$ and $(L\text{-}1)$ such that,

$$c_{ij}^L = w_{ij}^L y_j^{L-1} \qquad (2)$$

$\sigma(\cdot)$ is the standard sigmoidal nonlinearity, $y_i^{L-1}$ is the output of a node in layer $L\text{-}1$, $y_0^{L-1} = 1$, and the $w_{ij}^L$ are adjustable network weights.  Some typical linear connection functions are shown in Fig. 2.  $c_{10}^L$ corresponds to a threshold input.

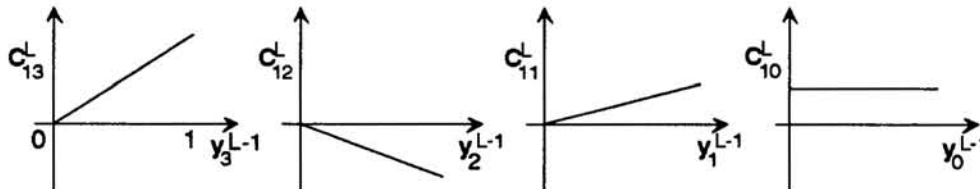

Figure 2.   Typical MLP Node Connection Functions

Incorporating B-Spline receptive field functions (Lane et al., 1991a) into the node computations of eq. (1) allows more general connection functions (e.g. piecewise linear, quadratic, cubic, etc.) to be formulated.  The corresponding B-Spline MLP (Spline Net) is derived by redefining the connection functions of eq. (2) such that,

$$c_{ij}^L(y_j^{L-1}) = \sum_{k} w_{ijk}^L B_{nk}^G(y_j^{L-1}) \qquad (3)$$

This enables the construction of a more general neural network architecture that has node equations of the form,

$$y_i^L = \sigma[\sum_{j=1}^{\eta_{L-1}} c_{ij}^L(y_j^{L-1})] = \frac{1}{1 + \exp[-\sum_{j=1}^{\eta_{L-1}} c_{ij}^L(y_j^{L-1})]} \qquad (4)$$

The $B_{nk}^{G}(y_{j}^{L-1})$ are B-spline receptive field functions (Lane et al, 1989,1991a) of order $n$ and support $G$, while the $w_{ijk}^{L}$ are the spline network weights. The order, $n$, corresponds to the number of coefficients in the polynomial pieces. For example, linear splines are of order $n=2$, whereas cubic splines are of order $n=4$. The advantage of the more general B-Spline connection functions of eq. (3) is that it allows varying degrees of "locality" to be added to the network computations since network weights are now activated based on the value of $y_{j}^{L-1}$. The $w_{ijk}^{L}$ are modified by backpropagating the output error only to the $G$ weights in each connection function associated with active (i.e. nonzero) receptive field functions. The $L^{\text{th}}$-layer weights are updated using the method of steepest descent learning such that,

$$w_{ijk}^{L} \leftarrow w_{ijk}^{L} + \beta e_{i}^{L} y_{i}^{L}(1 - y_{i}^{L})B_{nk}^{G}(y_{j}^{L-1}) \tag{5}$$

where $e_{i}^{L}$ is the output error back-propagated to the $i^{\text{th}}$ node in layer $L$ and $\beta$ is the learning rate (Lane et al., 1991a). In the more general Spline Net formulation of eqs. (3-5), each node input has $P+G-1$ receptive fields and $P+G-1$ weights associated with it, but only G are active at any one time. $P$ determines the number of partitions in the input space of the connection functions. Standard MLP networks are a degenerate case of the Spline Net architecture, as they can be realized with B-Spline receptive field functions of order $n=2$, with $P=1$ and $G=2$. Due to the connectivity of the B-Spline receptive field functions, for the case when $P>1$, the resulting network architecture corresponds to multiply-connected MLPs, where any given MLP is active within only one hypercube in the input space, but has weights that are shared with MLPs on the neighboring hypercubes. The amount of computation required in each layer of a Spline Net during both learning and function approximation is proportional to $G$, and independent of $P$.

Formulating the connection functions of eq. (3) with linear ($n=2$) B-Splines allows connection functions such as those shown in Fig. 3 to be learned.

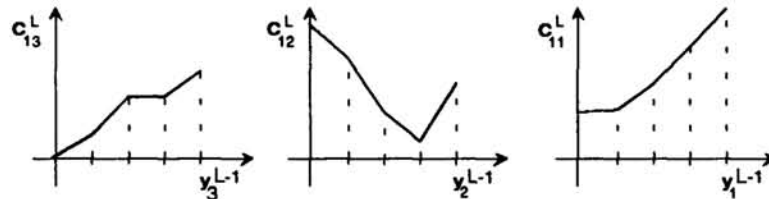

Figure 3.   Spline Net Connection Functions Using Linear B-Splines ($n=2$)

The connection functions shown in Fig. 3 have $P=4$ partitions (5 knots) on the interval $y_{j}^{L-1} \in [0,1]$. The number of input partitions, $P$, determines the degree of locality of

the resulting function approximation since the local shape of the connection function is determined from the current node input activation interval.

Networks constructed using the Spline Net formulation are reminiscent of the form and function of Kolmogorov-Lorenz networks (Baron and Baron, 1988). A neurobiological interpretation of a Spline Net is that it is composed of neurons that have dendritic branches with synapses that operate as a function of the level of activation at a given node or network input. This is shown in the network architecture of Fig. 4b where the standard three-layer MLP network of Fig. 4a has been redrawn using B-Spline receptive field functions with $n=2$, $P=4$ and $G=2$.

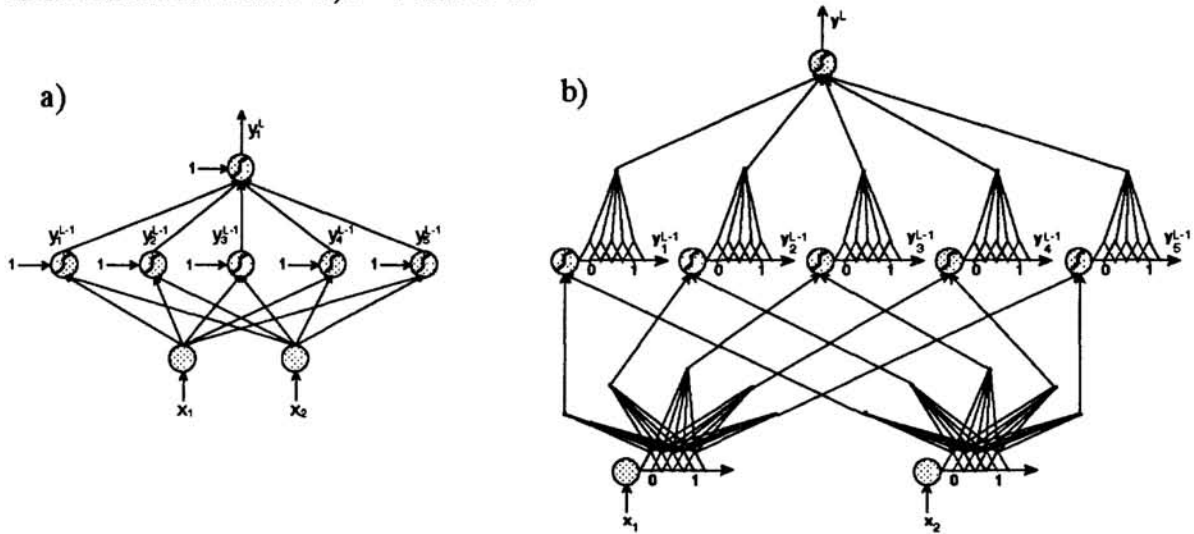

Figure 4.  Three-Layer Spline Net Architecture, $n=2,P=4,G=2$

The horizontal arrows projecting from the right of each network node in Fig. 4b represent the node outputs. The overlapping triangles on the node output represent the receptive field functions of neurons in the next layer. These receptive field functions are summed with weighted connections in the dendritic branches to form the inputs to the next network layer. In the architecture shown in Fig. 4b, only two receptive fields are active for any given value of a node output. Therefore for this single hidden-layer network architecture, given any value for the inputs $(x_1,x_2)$, at most $N_w = 30$ weights will be active where,

$$N_w = (s+1)\eta G \qquad (6)$$

$s$ is the number of network inputs and $\eta$ is the number of nodes in the hidden layer, which for this case is $2s+1 = 5$.

## 3.  SIMULATION RESULTS

In order to evaluate the impact of local computation on MLP performance, the well known spiral problem of Weiland and of Lang and Witbrock (1988) was chosen as a benchmark. Simulations were conducted using a Spline Net architecture having one hidden layer with 5 hidden nodes and linear B-Splines with support, $G=2$ (Fig. 4). All trials used the "vanilla" back-prop learning rule of eq. (5) with $\beta = 1/(2P)$. The connection function weights were initialized in each node such that the resulting connection functions were continuous linear functions with arbitrary slope. From previous experience (Lane et al., 1989), it was known that the number of receptive field partitions can drastically affect network learning and performance. Therefore, the connection function partitions were bifurcated during training to see the effect on network generalization capability and learning speed. The bifurcation consisted of splitting every receptive field in half after increments of 100K (100,000) training points, each time doubling the number of connection function partitions and weights in the network nodes. A more adaptive approach would monitor the slope of the learning curve to determine when to split the partitions. New weights were initializing such that the connection functions before and after the bifurcation retained the same shape. All simulation results presented in Figs. 5-12 were generated using 800K training points.

The left-most column of Fig. 5 represents the two learned connection functions that lead to each hidden node depicted in Fig. 4. The elements in the second column are the hidden node response to excitation over the unit square, while the plots in the third column are the connection functions from the hidden layer to the output node. The fourth column shows the hidden node outputs after being passed through their respective connection functions. The network output shown in the fifth column is the algebraic sum of the hidden node responses shown in the fourth column. The Spline Net was initialized as a standard MLP with $P=1$. Figure 6 shows the evolution of the two connection functions to the third hidden node in Fig. 4 after every 100K training points. Around 400K ($P=8$) the connection functions start to take on a characteristic shape. For $P>8$, the creation of additional partitions has little effect on the shape of the connection functions. Figure 7 shows the associated learning curve, while Fig. 8 is an enlarged version of the network output. These results indicate that the bifurcation schedule introduces additional degrees of freedom (weights) to the network in such a way as to carve out coarse global features first, then incrementally capture finer and finer localized details later. This is in contrast to the results shown in Figs. 9 and 10 where the training (using the same 800K points as in Figs. 7 and 8) was begun on a network having $P=128$ initial partitions. Figure 11 shows the Spline Net output after 800K training iterations using 112 discrete points located on the two spirals. Lang and Witbrock (1988) state that similar spiral results could only be obtained using a MLP network with 3 hidden layers (including jump connections) and 50,000,000 training iterations. The use of a Spline Net with a bifurcation schedule enabled the learning to be sped up by almost two orders of magnitude, indicating there is a significant performance advantage in trading-off number of hidden layers for node complexity.

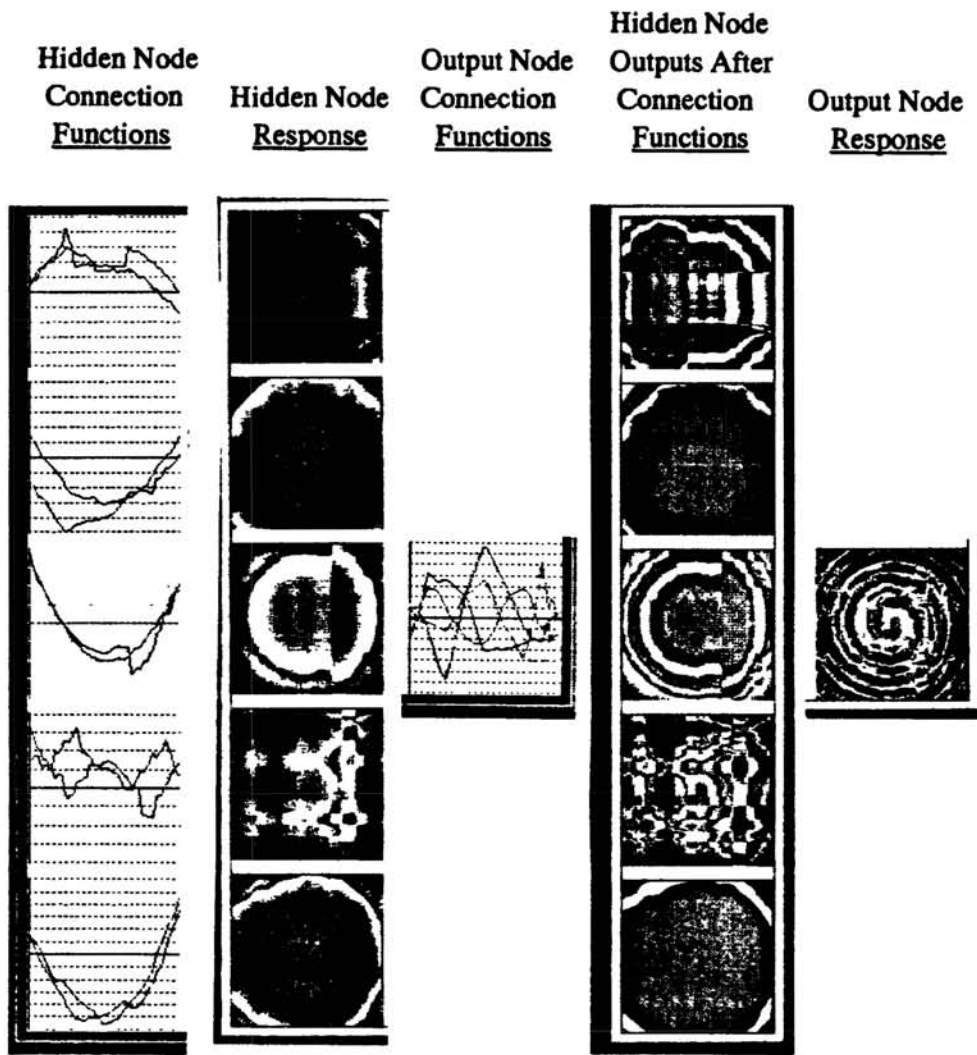

Figure 5.   Spiral Learning with Bifurcation Schedule

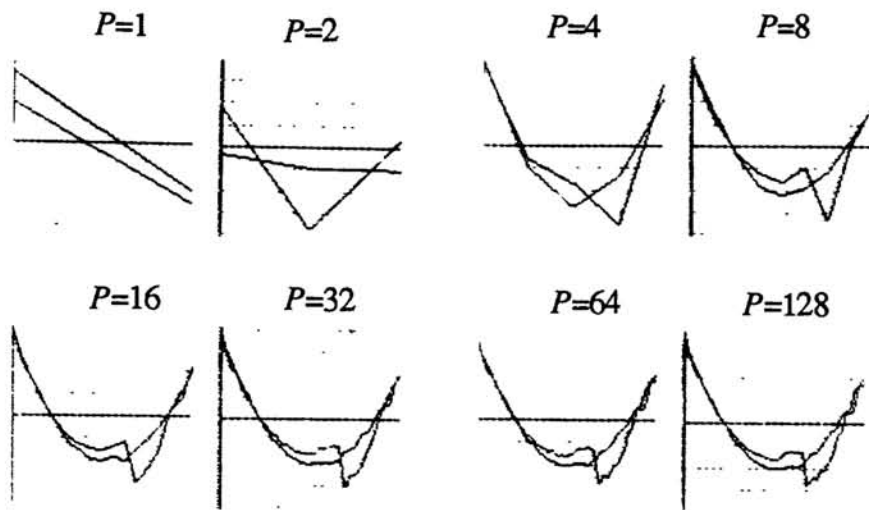

Figure 6.   Evolution of Connection Functions to Third Hidden Node

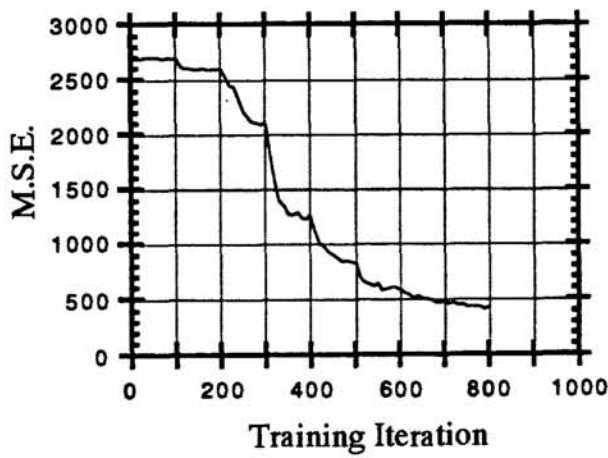

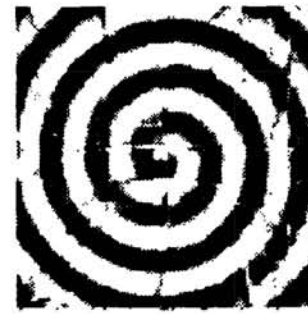

Figure 8.   Output Node Response

with Bifurcation

Figure 7.   Learning Curve with Bifurcation Schedule
Mean Square Error vs. Training Iteration

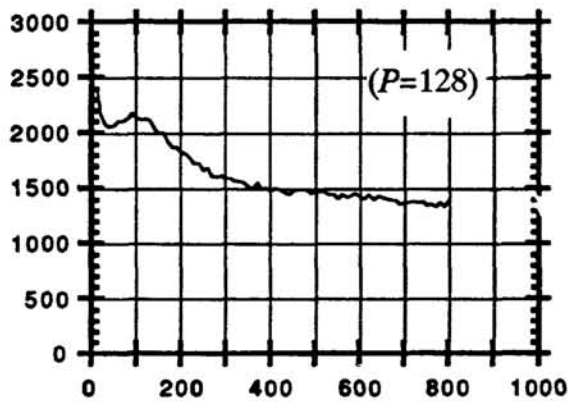

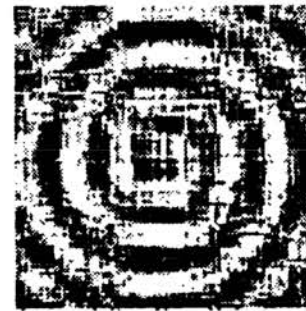

Figure 10.   Output Node Response

without Bifurcation

Figure 9.   Learning Curve without Bifurcation Schedule
Mean Square Error vs. Training Iteration

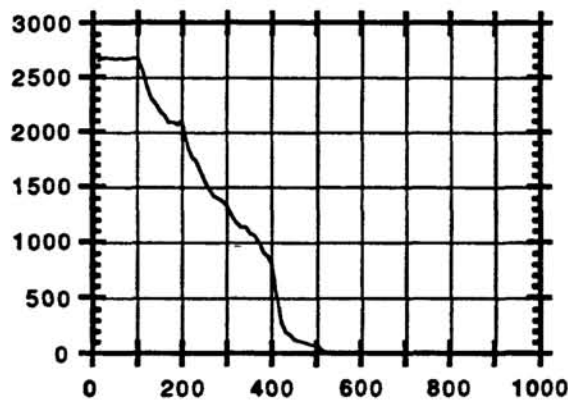

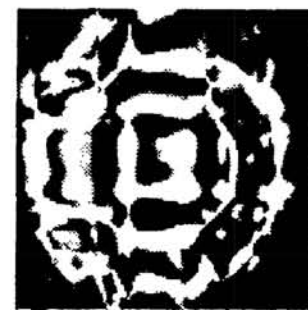

Figure 12. Output Node Response

with Bifurcation

(112 Discrete Points)

Figure 11. Learning Curve with Bifurcation Schedule
Mean Square Error vs. Training Iteration

(112 Discrete Points)

## 4.    CONCLUSIONS

It was shown that the introduction of B-Splines into the node connection functions of Multi-Layer Perceptrons allows more general neural network architectures to be developed. The resulting Spline Net architecture combines the fast learning and computational efficiency of strictly local neural network approaches with the scaling and generalization properties of the more established global MLP approach. Similarity to Kolmogorov-Lorenz networks can be used to suggest an initial number of hidden layer nodes. The number of node connection function partitions chosen affects both network generalization capability and learning performance. It was shown that use of a bifurcation schedule to determine the number of node input partitions speeds learning and improves network generalization. Results indicate that Spline Nets solve difficult learning problems by trading-off number of hidden layers for node complexity.

### Acknowledgements

Stephen H. Lane and David A. Handelman are also employed by Robicon Systems Inc., Princeton, NJ. This research has been supported through a grant from the James S. McDonnell Foundation and a contract from the DARPA Neural Network Program.

### References

Albus, J. (1975) "A New Approach to Manipulator Control: The Cerebellar Model Articulation Controller (CMAC)," *J. Dyn. Sys. Meas. Control*, vol. 97, pp. 270-277.

Barron, A.R. and Barron, R.L. (1988) "Statistical Learning Networks: A Unifying View," *Proc. 20th Symp. on the Interface - Computing and Statistics*, pp. 192-203.

Hornik, K. Stinchcombe, M. and White, H. (1989) "Multi-layer Feedforward Networks are Universal Approximators," *Neural Networks*, vol. 2, pp. 359-366.

Hubel, D. and Wiesel, T.N. (1962) "Receptive Fields, Binocular Interaction and Functional Architecture in Cat's Visual Cortex," *J. Physiology*, vol. 160, no. 106.

Lane, S.H., Handelman, D.A. and Gelfand, J.J. (1989) "Development of Adaptive B-Splines Using CMAC Neural Networks", *1989 IJCNN*, Wash. DC., June 1989.

Lane, S.H., Flax, M.B., Handelman, D.A. and Gelfand, J.J. (1991a) "Function Approximation in Multi-Layer Neural Networks with B-Spline Receptive Field Functions," Princeton University Cognitive Science Lab Report No. 42, in prep for *J. of Int'l Neural Network Society*.

Lane, S.H., Handelman, D.A. and Gelfand, J.J. (1991b) "Higher-Order CMAC Neural Networks-Theory and Practice," to appear Amer. Contr. Conf., Boston, MA, June,1991.

Lang, K.J. and Witbrock, M.J. (1988) "Learning to Tell Two Spirals Apart," *Proc. 1988 Connectionist Model Summer School*, D. Touretzky, G. Hinton, and T. Sejnowski, Eds.

Moody, J. and Darken, C. (1988) "Learning with Localized Receptive Fields," *Proc. 1988 Connectionist Model Summer School*, D. Touretzky, G. Hinton, T.Sejnowski, Eds.